# Modelling Genetic Variations with Fragmentation-Coagulation Processes

**Yee Whye Teh, Charles Blundell and Lloyd T. Elliott**
Gatsby Computational Neuroscience Unit, UCL
17 Queen Square, London WC1N 3AR, United Kingdom
{ywteh,c.blundell,elliott}@gatsby.ucl.ac.uk

## Abstract

We propose a novel class of Bayesian nonparametric models for sequential data called fragmentation-coagulation processes (FCPs). FCPs model a set of sequences using a partition-valued Markov process which evolves by splitting and merging clusters. An FCP is exchangeable, projective, stationary and reversible, and its equilibrium distributions are given by the Chinese restaurant process. As opposed to hidden Markov models, FCPs allow for flexible modelling of the number of clusters, and they avoid label switching non-identifiability problems. We develop an efficient Gibbs sampler for FCPs which uses uniformization and the forward-backward algorithm. Our development of FCPs is motivated by applications in population genetics, and we demonstrate the utility of FCPs on problems of genotype imputation with phased and unphased SNP data.

## 1  Introduction

We are interested in probablistic models for sequences arising from the study of genetic variations in a population of organisms (particularly humans). The most commonly studied class of genetic variations in humans are single nucleotide polymorphisms (SNPs), with large quantities of data now available (e.g. from the HapMap [1] and 1000 Genomes projects [2]). SNPs play an important role in our understanding of genetic processes, human historical migratory patterns, and in genome-wide association studies for discovering the genetic basis of diseases, which in turn are useful in clinical settings for diagnoses and treatment recommendations.

A SNP is a specific location in the genome where a mutation has occurred to a single nucleotide at some time during the evolutionary history of a species. Because the rate of such mutations is low in human populations the chances of two mutations occurring in the same location is small and so most SNPs have only two variants (wild type and mutant) in the population. The SNP variants on a chromosome of an individual form a sequence, called a haplotype, with each entry being binary valued coding for the two possible variants at that SNP. Due to the effects of gene conversion and recombination, the haplotypes of a set of individuals often has a "mosaic" structure where contiguous subsequences recur across multiple individuals [3]. Hidden Markov Models (HMMs) [4] are often used as the basis of existing models of genetic variations that exploit this mosaic structure (e.g. [3, 5]). However, HMMs, as dynamic generalisations of finite mixture models, cannot flexibly model the number of states needed for a particular dataset, and suffer from the same label switching non-identifiability problems of finite mixture models [6] (see Section 3.2). While nonparametric generalisations of HMMs [7, 8, 9] allow for flexible modelling of the number of states, they still suffer from label switching problems.

In this paper we propose alternative Bayesian nonparametric models for genetic variations called fragmentation-coagulation processes (FCPs). An FCP defines a Markov process on the space of partitions of haplotypes, such that the random partition at each time is marginally a Chinese restaurant

process (CRP). The clusters of the FCP are used in the place of HMM states. FCPs do not require the number of clusters in each partition to be specified, and do not have explicit labels for clusters thus avoid label switching problems. The partitions of FCPs evolve via a series of events, each of which involves either two clusters merging into one, or one cluster splitting into two. We will see that FCPs are natural models for the mosaic structure of SNP data since they can flexibly accommodate varying numbers of subsequences and they do not have the label switching problems inherent in HMMs. Further, computations in FCPs scale well.

There is a rich literature on modelling genetic variations. The standard coalescent with recombination (also known as the ancestral recombination graph) model describes the genealogical history of a set of haplotypes using coalescent, recombination and mutation events [10]. Though an accurate model of the genetic process, inference is unfortunately highly intractable. PHASE [11, 12] and IMPUTE [13] are a class of HMM based models, where each HMM state corresponds to a haplotype in a reference panel (training set). This alleviates the label switching problem, but incurs higher computational costs than the normal HMMs or our FCP since there are now as many HMM states as reference haplotypes. BEAGLE [14] introduces computational improvements by collapsing the multiple occurrences of the same mosaic subsequence across the reference haplotypes into a single node of a graph, with the graph constructed in a very efficient but somewhat ad hoc manner.

Section 2 introduces preliminary notation and describes random partitions and the CRP. In Section 3 we introduce FCPs, discuss their more salient properties, and describe how they are used to model SNP data. Section 4 describes an auxiliary variables Gibbs sampler for our model. Section 5 presents results on simulated and real data, and Section 6 concludes.

## 2 Random Partitions

Let $S$ denote a set of $n$ SNP sequences. Label the sequences by the integers $1, \ldots, n$ so that $S$ can be taken to be $[n] = \{1, \ldots, n\}$. A partition $\gamma$ of $S$ is a set of disjoint non-empty subsets of $S$ (called clusters) whose union is $S$. Denote the set of partitions of $S$ by $\mathbf{\Pi}_S$. If $a \subset S$, define the projection $\gamma_{|a}$ of $\gamma$ onto $a$ to be the partition of $a$ obtained by removing the elements of $S \backslash a$ as well as any resulting empty subsets from $\gamma$. The canonical distribution over $\mathbf{\Pi}_S$ is the Chinese restaurant process (CRP) [15, 16]. It can be described using an iterative generative process: $n$ customers enter a Chinese restaurant one at a time. The first customer sits at some table and each subsequent customer sit at a table with $m$ current customers with probability proportional to $m$, or at a new table with probability proportional to $\alpha$, where $\alpha$ is a parameter of the CRP. The seating arrangement of customers around tables forms a partition $\gamma$ of $S$, with occupied tables corresponding to the clusters in $\gamma$. We write $\gamma \sim \mathrm{CRP}(\alpha, S)$ if $\gamma \in \mathbf{\Pi}_S$ is a CRP distributed random partition over $S$. Multiplying the conditional probabilities together gives the probability mass function of the CRP:

$$f_{\alpha,S}(\gamma) = \frac{\alpha^{|\gamma|}\Gamma(\alpha)}{\Gamma(n+\alpha)} \prod_{a \in \gamma} \Gamma(|a|) \tag{1}$$

where $\Gamma$ is the gamma function. The CRP is exchangeable (invariant to permutations of $S$), and projective (the probability of the projection $\gamma_{|a}$ is simply $f_{\alpha,a}(\gamma_{|a})$), so can be extended in a natural manner to partitions of $\mathbb{N}$ and is related via de Finetti's theorem to the Dirichlet process [17].

## 3 Fragmentation-Coagulation Processes

A fragmentation-coagulation process (FCP) is a continuous-time Markov process $\pi \equiv (\pi(t), t \in [0, T])$ over a time interval $[0, T]$ where each $\pi(t)$ is a random partition in $\mathbf{\Pi}_S$. Since the space of partitions for a finite $S$ is finite, the FCP is a Markov jump process (MJP) [18] : it evolves according to a discrete series of random events (or jumps) at which it changes state and at all other times the state remains unchanged. In particular, the jump events in an FCP are either fragmentations or coagulations. A fragmentation at time $t$ involves a cluster $c \in \pi(t-)$ splitting into exactly two non-empty clusters $a, b \in \pi(t)$ (all other clusters stay unchanged; the $t-$ notation means an infinitesimal time before $t$), and a coagulation at $t$ involves two clusters $a, b \in \pi(t-)$ merging to form a single cluster $c = a \cup b \in \pi(t)$ (see Figure 1). Note that fragmentations and coagulations are converses of each other; as we will see later, this will lead to some important properties of the FCP.

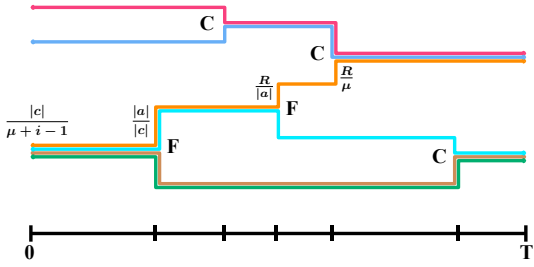

Figure 1: FCP cartoon. Each line is a sequence and bundled lines form clusters. C: coagulation event. F: fragmentation event. Fractions are, for the orange sequence, from left to right: probability of joining cluster $c$ at time 0, probability of following cluster $a$ at a fragmentation event, rate of starting a new table (creating a fragmentation), and rate of joining with an existing table (creating a coagulation).

Following the various popular culinary processes in Bayesian nonparametrics, we will start by describing the law of $\pi$ in terms of the conditional distribution of the cluster membership of each sequence $i$ given those of $1, \dots, i-1$. Since we have a Markov process with a time index, the metaphor is of a Chinese restaurant operating from time 0 to time $T$, where customers (sequences) may move from one table (cluster) to another and tables may split and merge at different points in time, so that the seating arrangements (partition structures) at different times might not be the same. To be more precise, define $\pi_{|[i-1]} = (\pi_{|[i-1]}(t), t \in [0,T])$ to be the projection of $\pi$ onto the first $i-1$ sequences. $\pi_{|[i-1]}$ is piecewise constant, with $\pi_{|[i-1]}(t) \in \mathbf{\Pi}_{[i-1]}$ describing the partitioning of the sequences $1, \dots, i-1$ (the seating arrangement of customers $1, \dots, i-1$) at time $t$. Let $a_i(t) = c \backslash \{i\}$, where $c$ is the unique cluster in $\pi_{|[i]}(t)$ containing $i$. Note that either $a_i(t) \in \pi_{|[i-1]}(t)$, meaning customer $i$ sits at an existing table in $\pi_{|[i-1]}(t)$, or $a_i(t) = \emptyset$, which will mean that customer $i$ sits at a new table. Thus the function $a_i$ describes customer $i$'s choice of table to sit at through times $[0,T]$. We define the conditional distribution of $a_i$ given $\pi_{|[i-1]}$ as a Markov jump process evolving from time 0 to $T$ with two parameters $\mu > 0$ and $R > 0$ (see Figure 1):

$i = 1$: The first customer sits at a table for the duration of the process, i.e. $a_1(t) = \emptyset \ \forall t \in [0,T]$.

$t = 0$: Each subsequent customer $i$ starts at time $t = 0$ by sitting at a table according to CRP probabilities with parameter $\mu$. So, $a_i(0) = c \in \pi_{|[i-1]}(0)$ with probability proportional to $|c|$, and $a_i(0) = \emptyset$ with probability proportional to $\mu$.

F1: At time $t > 0$, if customer $i$ is sitting at table $a_i(t-) = c \in \pi_{|[i-1]}(t-)$, and the table $c$ fragments into two tables $a, b \in \pi_{|[i-1]}(t)$, customer $i$ will move to table $a$ with probability $|a|/|c|$, and to table $b$ with probability $|b|/|c|$.

C1: If the table $c$ merges with another table at time $t$, the customer simply follows the other customers to the resulting merged table.

F2: At all other times $t$, if customer $i$ is sitting at some existing table $a_i(t-) = c \in \pi_{|[i-1]}(t)$, then the customer will move to a new empty table ($a_i(t) = \emptyset$) with rate $R/|c|$.

C2: Finally, if $i$ is sitting by himself ($a_i(t-) = \emptyset$), then he will join an existing table $a_i(t) = c \in \pi_{|[i-1]}(t)$ with rate $R/\mu$. The total rate of joining any existing table is $|\pi_{|[i-1]}(t)|R/\mu$.

Note that when customer $i$ moves to a new table in step F2, a fragmentation event is created, and all subsequent customers who end up in the same table will have to decide at step F1 whether to move to the original table or to the table newly created by $i$. The probabilities in steps F1 and F2 are exactly the same as those for a Dirichlet diffusion tree [19] with constant divergence function $R$. Similarly step C2 creates a coagulation event in which subsequent customers seated at the two merging tables will move to the merged table in step C1, and the probabilities are exactly the same as those for Kingman's coalescent [20, 21]. Thus our FCP is a combination of the Dirichlet diffusion tree and Kingman's coalescent. Theorem 3 below shows that this combination results in FCPs being stationary Markov processes with CRP equilibrium distributions. Further, FCPs are reversible, so in a sense the Dirichlet diffusion tree and Kingman's coalescent are duals of each other.

Given $\pi_{|[i-1]}$, $\pi_{|[i]}$ is uniquely determined by $a_i$ and vice versa, so that the seating of all $n$ customers through times $[0,T]$, $a_1, \dots, a_n$, uniquely determines the sequential partition structure $\pi$. We now investigate various properties of $\pi$ that follows from the iterative construction above. The first is an alternative characterisation of $\pi$ as an MJP whose transitions are fragmentations or coagulations, an unsurprising observation since both the Dirichlet diffusion tree and Kingman's coalescent, as partition-valued processes, are Markov.

**Theorem 1.** *$\pi$ is an MJP with initial distribution $\pi(0) \sim \mathrm{CRP}(\mu, S)$ and stationary transit rates,*

$$q(\gamma, \rho) = R \frac{\Gamma(|a|)\Gamma(|b|)}{\Gamma(|c|)} \qquad\qquad q(\rho, \gamma) = \frac{R}{\mu} \tag{2}$$

*where $\gamma, \rho \in \mathbf{\Pi}_S$ are such that $\rho$ is obtained from $\gamma$ by fragmenting a cluster $c \in \gamma$ into two clusters $a, b \in \rho$ (at rate $q(\gamma, \rho)$), and conversely $\gamma$ is obtained from $\rho$ by coagulating $a, b$ into $c$ (at rate $q(\rho, \gamma)$). The total rate of transition out of $\gamma$ is:*

$$q(\gamma, \cdot) = R \sum_{c \in \gamma} H_{|c|-1} + \frac{R}{\mu} \frac{|\gamma|(|\gamma|-1)}{2} \tag{3}$$

*where $H_{|c|-1}$ is the $|c|-1$st harmonic number.*

*Proof.* The initial distribution follows from the CRP probabilities of step $t = 0$. For every $i$, $a_i$ is Markov and $a_i(t)$ depends only on $a_i(t-)$ and $\pi_{|[i-1]}(t)$, thus $(a_i(s), s \in [0, t])$ depends only on $(a_j(s), s \in [0, t], j < i)$ and the Markovian structure of $\pi$ follows by induction. Since $\mathbf{\Pi}_S$ is finite, $\pi$ is an MJP. Further, the probabilities and rates in steps F1, F2, C1 and C2 do not depend explicitly on $t$ so $\pi$ has stationary transit rates. By construction, $q(\gamma, \rho)$ is only non-zero if $\gamma$ and $\rho$ are related by a complimentary pair of fragmentation and coagulation events, as in the theorem.

To derive the transition rates (2), recall that a transition rate $r$ from state $s$ to state $s'$ means that if the MJP is in state $s$ at time $t$ then it will transit to state $s'$ by an infinitesimal time later $t + \delta$ with probability $\delta r$. For the fragmentation rate $q(\gamma, \rho)$, the probability of transiting from $\gamma$ to $\rho$ in an infinitesimal time $\delta$ is $\delta$ times the rate at which a customer starts his own table in step F2, times the probabilities of subsequent customers choosing either table in step F1 to form the two tables $a$ and $b$. Dividing this product by $\delta$ forms the rate $q(\gamma, \rho)$. Without loss of generality suppose that the table started by the customer eventually becomes $a$ and that there were $j$ other customers at the existing table which eventually becomes $b$. Thus, the rate of the customer starting his own table is $R/j$ and the product of probabilities of subsequent customer choices in step F1 is then $\frac{1 \cdot 2 \cdots (|a|-1) \times j \cdots (|b|-1)}{(j+1) \cdots (|c|-1)}$. Multiplying these together gives $q(\gamma, \rho)$ in (2). Similarly, the coagulation rate $q(\rho, \gamma)$ is a product of the rate $\frac{R}{\mu}$ at which a customer moves from his own table to an existing table in step C2 and the probability of all subsequent customers in either table moving to the merged table (which is just 1).

Finally, the total transition rate $q(\gamma, \cdot)$ is a sum over all possible fragmentations and coagulations of $\gamma$. There are $\frac{|\gamma|(|\gamma|-1)}{2}$ possible pairs of clusters to coagulate, giving the second term. The first term is obtained by summing over all $c \in \gamma$, and over all unordered pairs $a, b$ resulting from fragmenting $c$, and using the identity $\sum_{\{a,b\}} \frac{\Gamma(|a|)\Gamma(|b|)}{\Gamma(|c|)} = H_{|c|-1}$. $\qquad\square$

**Theorem 2.** *$\pi$ is projective and exchangeable. Thus it can be extended naturally to a Markov process over partitions of $\mathbb{N}$.*

*Proof.* Both properties follow from the fact that both the initial distribution $\mathrm{CRP}(\mu, S)$ and the transition rates (2) are projective and exchangeable. Here we will give more direct arguments for the theorem. Projectivity is a direct consequence of the iterative construction, showing that the law of $\pi_{|[i]}$ does not depend on the clustering trajectories $a_j$ of subsequent customers $j > i$. We can show exchangeability of $\pi$ by deriving the joint probability density of a sample path of $\pi$ (the density exists since both $\mathbf{\Pi}_S$ and $T$ are finite so $\pi$ has a finite number of events on $[0, T]$), and seeing that it is invariant to permutations of $S$. For an MJP the probability of a sample path is the probability of the initial state ($f_{\mu, S}(\pi(0))$) times, for each subsequent jump, the probability of staying in the current state $\gamma$ until the jump (the holding time is exponential distributed with rate $q(\gamma, \cdot)$) and the transition from $\gamma$ to the next state $\rho$ (this is the ratio $q(\gamma, \rho)/q(\gamma, \cdot)$), and finally the probability of not transiting from the last jump time to $T$. Multiplying these probabilities together gives, after simplification:

$$p(\pi) = R^{|C|+|F|} \mu^{|A|-2|C|-2|F|} \frac{\Gamma(\mu)}{\Gamma(\mu+n)} \exp\left( -\int_0^T q(\pi(t), \cdot)dt \right) \frac{\prod_{a \in A_{<>}} \Gamma(|a|)}{\prod_{a \in A_{><}} \Gamma(|a|)} \tag{4}$$

with $|C|$ the number of coagulations, $|F|$ number of fragmentations, and $A, A_{<>}, A_{><}$ are sets of paths in $\pi$. A path is a cluster created either at time 0 or a coagulation or fragmentation, and exists for a definite amount of time until it is terminated at time $T$ or another event (these are the horizontal

bundles of lines in Figure 1). $A$ is the set of all paths in $\pi$, $A_{<>}$ the set of paths created either at time 0 or by a fragmentation and terminated either at time $T$ or by a coagulation, and $A_{><}$ the set of paths created by a coagulation and terminated by a fragmentation or at time $T$. □

**Theorem 3.** $\pi$ *is ergodic and has equilibrium distribution* $\mathrm{CRP}(\mu, S)$. *Further, it is reversible with* $(\pi(T-t), t \in [0, T])$ *having the same law as* $\pi$.

*Proof.* Ergodicity follows from the fact that for any $T > 0$ and any two partitions $\gamma, \rho \in \mathbf{\Pi}_S$, there is positive probability that if it starts at $\pi(0) = \gamma$, it will end with $\pi(T) = \rho$. For example, it may undergo a sequence of fragmentations until each sequence belong to its own cluster, then a sequence of coagulations forming the clusters in $\rho$. Reversibility and the equilibrium distribution can be demonstrated by detailed balance. Suppose $\gamma, \rho \in \mathbf{\Pi}_S$ and $a, b, c$ are related as in Theorem 1,

$$f_{\mu,S}(\gamma)q(\gamma,\rho) = \frac{\mu^{|\gamma|}\Gamma(\mu)}{\Gamma(n+\mu)} \prod_{k\in\gamma} \Gamma(|k|) \times R\frac{\Gamma(|a|)\Gamma(|b|)}{\Gamma(|c|)} \tag{5}$$

$$= \frac{\mu^{|\gamma|+1}\Gamma(\mu)}{\Gamma(n+\mu)}\Gamma(|a|)\Gamma(|b|) \prod_{k\in\gamma, k\neq c} \Gamma(|k|) \times \frac{R}{\mu} = f_{\mu,S}(\rho)q(\rho,\gamma)$$

Finally, the terms in (4) are invariant to time reversals, i.e. $p((\pi(T-t), t \in [0, T])) = p(\pi)$. □

Theorem 3 shows that the $\mu$ parameter controls the marginal distributions of $\pi(t)$, while (2) indicates that the $R$ parameter controls the rate at which $\pi$ evolves.

## 3.1 A Model of SNP Sequences

We model the $n$ SNP sequences (haplotypes) with an FCP $\pi$ over partitions of $S = [n]$. Let the $m$ assayed SNP locations on a chunk of the chromosome be at positions $t_1 < t_2 \cdots < t_m$. The $i$th haplotype consists of observations $x_{i1}, \ldots, x_{im} \in \{0, 1\}$ each corresponding to a binary SNP variant. For $j = 1, \ldots, m$, and for each cluster $c \in \pi(t_j)$ at position $t_j$, we have a parameter $\theta_{cj} \sim \mathrm{Bernoulli}(\beta_j)$ which denotes the variant at location $t_j$ of the corresponding subsequence. For each $i \in c$ we model $x_{ij}$ as equal to $\theta_{cj}$ with probability $1 - \epsilon$, where $\epsilon$ is a noise probability. We place a prior $\beta_j \sim \mathrm{Beta}(\alpha\tilde{\beta}_j, \alpha(1 - \tilde{\beta}_j))$ with mean $\tilde{\beta}_j$ given by the empirical mean of variant 1 at SNP $j$ among the observed haplotypes. We place uninformative uniform priors on $\log R, \log \mu$ and $\log \alpha$ over a bounded but large range such that the boundaries were never encountered.

The properties of FCPs in Theorems 1-3 are natural in the modelling setting here. Projectivity and exchangeability relate to the assumption that sequence labels should not have an effect on the model, while stationarity and reversibility arise from the simplifying assumption that we do not expect the genetic processes operating in different parts of the genome to be different. These are also properties of the standard coalescent with recombination model of genetic variations [10]. Incidentally the coalescent with recombination model is not Markov, though there have been Markov approximations [22, 23], and all practical HMM based methods are Markov.

## 3.2 HMMs and the Label Switching Problem

HMMs can also be interpreted as sequential partitioning processes in which each state at time step $t$ corresponds to a cluster in the partition at $t$. Since each sequence can be in different states at different times this automatically induces a partition-structured Markov process, where each partition consists of at most $K$ clusters ($K$ being the number of states in the HMM), and where each cluster is labelled with an HMM state. This labelling of the clusters in HMMs is a significant, but subtle, difference between HMMs and FCPs. Note that the clusters in FCPs are unlabelled, and defined purely in terms of the sequences they contain. This labelling of the clusters in HMMs are a significant source of non-identifiability in HMMs, since the likelihoods of data items (and often even the priors over transition probabilities) are invariant to the labels themselves so that each permutation over labels creates a mode in the posterior. This is the so called "label switching problem" for finite mixture models [6]. Since the FCP clusters are unlabelled they do not suffer from label switching problems. On the other hand, by having labelled clusters HMMs can share statistical strength among clusters across time steps (e.g. by enforcing the same emission probabilities from each cluster across time), while FCPs do not have a natural way of sharing statistical strength across time. This means that FCPs are not suitable for sequential data where there is no natural correspondence between times across different sequences, e.g. time series data like speech and video.

## 3.3 Discrete Time Markov Chain Construction

FCPs can be derived as continuous time limits of discrete time Markov chains constructed from fragmentation and coagulation operators [24]. This construction is more intuitive but lacks the rigour of the development described here. Let $\mathrm{CRP}(\alpha, d, S)$ be a generalisation of the CRP on $S$ with an additional discount parameter $d$ (see [25] for details). For any $\delta > 0$, construct a Markov chain over $\pi(0), \pi(\delta), \pi(2\delta), \dots$ as follows: $\pi(0) \sim \mathrm{CRP}(\mu, 0, S)$; then for every $m \geq 1$, define $\rho(m\delta)$ to be the partition obtained by fragmenting each cluster $c \in \pi((m-1)\delta)$ by a partition drawn independently from $\mathrm{CRP}(0, R\delta, c)$, and $\pi(m\delta)$ is constructed by coagulating into one the clusters of $\rho(m\delta)$ belonging to the same cluster in a draw from $\mathrm{CRP}(\mu/R\delta, 0, \rho(m\delta))$. Results from [26] (see also [27]) show that marginally each $\rho(m\delta) \sim \mathrm{CRP}(\mu, R\delta, S)$ and $\pi(m\delta) \sim \mathrm{CRP}(\mu, 0, S)$. The various properties of FCPs, i.e. Markov, projectivity, exchangeability, stationarity, and reversibility, hold for this discrete time Markov chain, and the continuous time $\pi$ can be derived by taking $\delta \to 0$.

# 4 Gibbs Sampling using Uniformization

We use a Gibbs sampler for inference in the FCP given SNP haplotype data. Each iteration of the sampler involves treating the $i$th haplotype sequence as the last sequence to be added into the FCP partition structure (making use of exchangeability), so that the iterative procedure described in Section 3 gives the conditional prior of $a_i$ given $\pi_{|S\setminus\{i\}}$. Coupling with the likelihood terms of $x_{i1}, \dots, x_{im}$ gives us the desired conditional distribution of $a_i$. Since this conditional distribution of $a_i$ is Markov, we can make use of the forward filtering-backward sampling procedure to sample it. However, $a_i$ is a continuous-time MJP so a direct application of the typical forward-backward algorithm is not possible. One possibility is to marginalise out the sample path of $a_i$ except at a finite number of locations (corresponding to the jumps in $\pi_{|S\setminus\{i\}}$ and the SNP locations). This approach is computationally expensive as it requires many matrix exponentiations, and does not resolve the issue of obtaining a full sample path of $a_i$, which may involve jumps at random locations we have marginalised out.

Instead, we make use of a recently developed MCMC inference method for MJPs [28]. This sampler introduces as auxiliary variables a set of "potential jump points" distributed according to a Poisson process with piecewise constant rates, such that conditioned on them the posterior of $a_i$ becomes a Markov chain that can only transition at either its previous jump locations or the potential jump points, and we can then apply standard forward-backward to sample $a_i$. For each $t$ the state space of $a_i(t)$ is $C_{it} \equiv \pi_{|S\setminus\{i\}} \cup \{\emptyset\}$. For $s, s' \in C_{it}$ let $Q_t(s, s')$ be the transition rate from state $s$ to $s'$ given in Section 3, with $Q_t(s, s) = -\sum_{s' \neq s} Q_t(s, s')$. Let $\Omega_t > \max_{s \in C_{it}} -Q_t(s, s)$ be an upper bound on the transition rates of $a_i$ at time $t$, $a_i'$ be the previous sample path of $a_i$, $J'$ be the jumps in $a_i'$, and $E$ consists of the $m$ SNP locations and the event times in $\pi_{|S\setminus\{i\}}$. Let $M_t(s)$ be the forward message at time $t$ and state $s \in C_{it}$. The resulting forward-backward sampling algorithm is given below. In addition we update the logarithms of $R, \mu$ and $\alpha$ by slice sampling.

1. Sample potential jumps $J^{\mathrm{aux}} \sim \mathrm{Poisson}(\Lambda)$ with rate $\Lambda(t) = \Omega_t + Q_t(a_i'(t), a_i'(t))$.
2. Compute forward messages by iterating over $t \in \{0\} \cup J^{\mathrm{aux}} \cup J' \cup E$ from left to right:
   2a. At $t = 0$, set $M_t(s) \propto |s|$ for $s \in \pi_{|S\setminus\{i\}}$ and $M_t(\emptyset) \propto \mu$.
   2b. At a fragmentation in $\pi_{|S\setminus\{i\}}$, say of $c$ into $a, b$, set $M_t(a) = \frac{|a|}{|c|}M_{t-}(c)$, $M_t(b) = \frac{|b|}{|c|}M_{t-}(c)$, and $M_t(k) = M_{t-}(k)$ for $k \neq a, b, c$. Here $t-$ denotes the time of the previous iteration.
   2c. At a coagulation in $\pi_{|S\setminus\{i\}}$, say of $a, b$ into $c$, set $M_t(c) = M_{t-}(a) + M_{t-}(b)$.
   2d. At an observation, say $t = t_j$, set $M_t(s) = p(x_{ij}|\theta_{sj})M_{t-}(s)$. We integrate out $\theta_{\emptyset j}$ and $\beta_j$.
   2e. At a potential jump in $J^{\mathrm{aux}} \cup J'$, set $M_t(s) = \sum_{s' \in C_{it}} M_{t-}(s')(\mathbf{1}(s' = s) + Q_t(s', s)/\Omega)$.
3. Get new sample path $a_i$ by backward sampling. This is straightforward and involves reversing the message computations above. Note that $a_i$ can only jump at the times in $J^{\mathrm{aux}} \cup J'$, and change state at times in $E$ if it was involved in the fragmentation or coagulation event.

# 5 Experiments

**Label switching problem** Figure 2 demonstrates the label switching problem (Section 3.2) during block Gibbs sampling of a 2-state Bayesian HMM (BHMM) compared to inference in an FCP. The

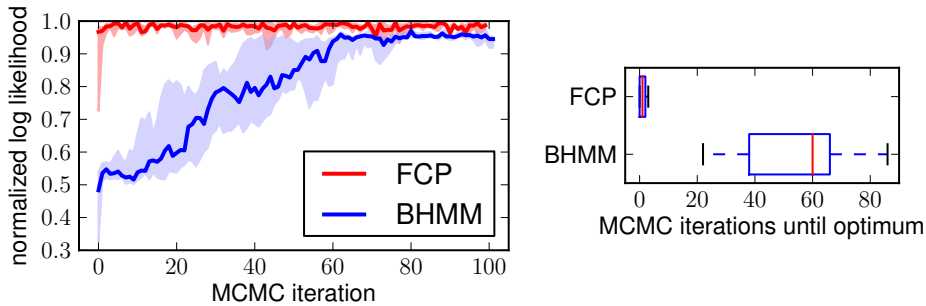

Figure 2: Label switching problem.
*Left:* Each line is median, over 10 runs, of the normalized log-likelihoods of a Bayesian HMM (blue) and an FCP (red) at each iteration of MCMC. Lighter polygons are the 25% and 75% percentiles.
*Right:* Number of MCMC iterations before each model first encounters the optimum states.

observed data comprises 16 sequences of length 16. Eight of the sequences consist of just zeros and the others consist of just ones. Each of the binary BHMM states, $z_{ij} \in \{0, 1\}$, $i$ indexing sequence and $j$ indexing position within sequence $i$, transits to the same state with probability $\tau$, with a prior $\tau \sim \text{Beta}(10.0, 0.1)$ encouraging self transitions. The observations of the BHMM have distribution $x_{ij} \sim \text{Bernoulli}(\rho_{z_{ij}})$ where $\rho_1 = 1 - \rho_0$ and $\rho_0 \sim \text{Beta}(1.0, 1.0)$. The optimal clustering under both models assigns all zero observations to one state and all ones to another state. As shown in Figure 2, due to the lack of identifiability of its states, the BHMM requires more MCMC iterations through the data before inference converges upon an optimal state, whilst an FCP is able to find the correct state much more quickly. This is reflected in both the normalized log-likelihood of the models in Figure 2(left) and in the number of iterations before reaching the optimal state, Figure 2(right).

**Imputation from phased data** To reduce costs, typically not all known SNPs are assayed for each participant in a large association study. The problem of inferring the variants of unassayed SNPs in a study using a larger dataset (e.g. HapMap or 1000 Genomes) is called genotype imputation [13].

Figure 3 compares the genotype imputation accuracy of FCP with that of fastPHASE [5] and BEAGLE [14], two state-of-the-art methods. We used 3000 MCMC iterations for inference with the FCP, with the first 1000 iterations discarded as burn-in. We used 320 genes from 47 individuals in the Seattle SNPs dataset [29]. Each gene consists of 94 sequences, of length between 13 and 416 SNPs. We held out 10%–50% of the SNPs uniformly among all haplotypes for testing. Our model had higher accuracy than both fastPHASE and BEAGLE.

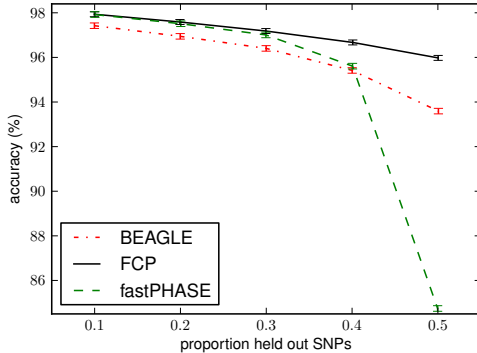

Figure 3: Accuracy vs proportion of missing data for imputation from phased data. Lines are drawn at the means and error bars at the standard error of the means.

**Imputation from unphased data** In humans, most chromosomes come in pairs. Current assaying methods are unable to determine from which of these two chromosomes each variant originates without employing expensive protocols, thus the data for each individual in large datasets actually consist of sequences of unordered pairs of variants (called genotypes). This includes the Seattle SNPs dataset (the haplotypes provided by [29] in the previous experiment were phased using PHASE [11, 12]).

In this experiment, we performed imputation using the original unphased genotypes, using an extension of the FCP able to handle this sort of data. Figure 4 shows the genotype imputation accuracies and run-times of the FCP model (with 60, 600 or 3000 MCMC iterations of which 30, 200 or 600 were discarded for burn-in) and state-of-the-art software (fastPHASE [5], IMPUTE2 [30], BEAGLE

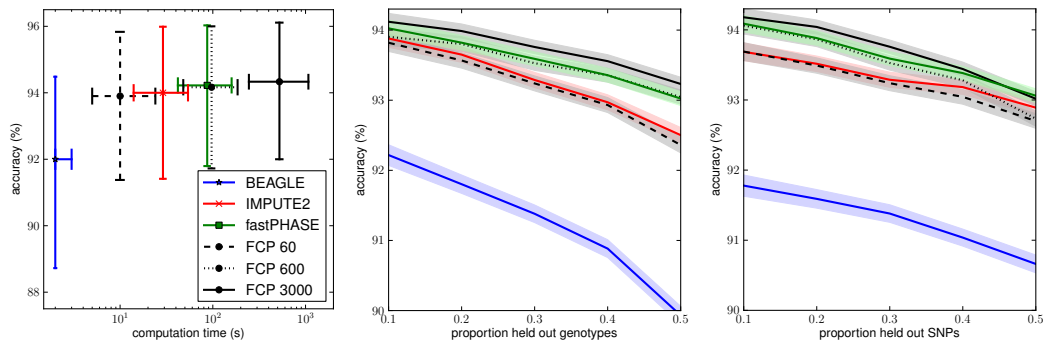

Figure 4: Time and accuracy performance of genotype imputation on 231 Seattle SNPs genes.
*Left:* Accuracies evaluated by removing 10%–50% of SNPs from 10%–50% of individuals, repeated five times on each gene with the same hold out proportions. Centers of crosses correspond to median accuracy and times whilst whiskers correspond to the extent of the inter-quartile range.
*Middle:* Lines are accuracy averaged over five repetitions of each gene with 30% of shared SNPs removed from 10%–50% of individuals. Each repetition uses a different subset of SNPs and individuals. Lighter polygons are standard errors.
*Right:* As Middle, except with 10%–50% of shared SNPs removed from 30% of individuals.

[14]). We held out 10%–50% of the shared SNPs in 10%–50% of the 47 individuals of the Seattle SNPs dataset. This paradigm mimics a popular experimental setting in which the genotypes of sparsely assayed individuals are imputed using a densely assayed reference panel [30]. We discarded 89 of the genes as they were unable to be properly pre-processed for use with IMPUTE2.

As can be seen in Figure 4, FCP achieves similar state-of-the-art accuracy to IMPUTE2 and fast-PHASE. Given enough iterations, the FCP outperforms all other methods in terms of accuracy. With 600 iterations, FCP has almost the same accuracy and run-time as fastPHASE. With just 60 iterations, FCP performs comparably to IMPUTE2 but is an order of magnitude faster. Note that IMPUTE2 scales quadratically in the number of genotypes, so we expect FCPs to be more scalable. Finally, BEAGLE is the fastest algorithm but has worst accuracies.

# 6 Discussion

We have proposed a novel class of Bayesian nonparametric models called fragmentation-coagulation processes (FCPs), and applied them to modelling population genetic variations, showing encouraging empirical results on genotype imputation. FCPs are the simplest non-trivial examples of exchangeable fragmentation-coalescence processes (EFCP) [31]. In general EFCPs the fragmentation and coagulation events may involve more than two clusters. They also have an erosion operation, where a single element of $S$ forms a single element cluster. EFCPs were studied by probabilists for their theoretical properties, and our work represents the first application of EFCPs as probabilistic models of real data, and the first inference algorithm derived for EFCPs.

There are many interesting avenues for future research. Firstly, we are currently exploring a number of other applications in population genetics, including phasing and genome-wide association studies. Secondly, it would be interesting to explore the discrete time Markov chain version of FCPs, which although not as elegant will have simpler and more scalable inference. Thirdly, the haplotype graph in BEAGLE is constructed via a series of cluster splits and merges, and bears striking resemblance to the partition structures inferred by FCPs. It would be interesting to explore the use of BEAGLE as a fast initialisation of FCPs, and to use FCPs as a Bayesian interpretation of BEAGLE. Finally, beyond population genetics, FCPs can also be applied to other time series and sequential data, e.g. the time evolution of community structure in network data, or topical change in document corpora.

### Acknowledgements

We thank the Gatsby Charitable Foundation for generous funding, and Vinayak Rao, Andriy Mnih, Chris Holmes and Gil McVean for fruitful discussions.

# References

[1] The International HapMap Consortium. The international HapMap project. *Nature*, 426:789–796, 2003.

[2] The 1000 Genomes Project Consortium. A map of human genome variation from population-scale sequencing. *Nature*, 467:1061–1073, 2010.

[3] M. J. Daly, J. D. Rioux, S. F. Schaffner, T. J. Hudson, and R. S. Lander. High-resolution haplotype structure in the human genome. *Nature Genetics*, 29:229–232, 2001.

[4] L. Rabiner. A tutorial on hidden Markov models and selected applications in speech recognition. *Proceedings of the IEEE*, 77:257–285, 1989.

[5] P. Scheet and M. Stephens. A fast and flexible statistical model for large-scale population genotype data: Applications to inferring missing genotypes and haplotypic phase. *The American Journal of Human Genetics*, 78(4):629 – 644, 2006.

[6] A. Jasra, C. C. Holmes, and D. A. Stephens. Markov chain Monte Carlo methods and the label switching problem in Bayesian mixture modeling. *Statistical Science*, 20(1):50–67, 2005.

[7] M. J. Beal, Z. Ghahramani, and C. E. Rasmussen. The infinite hidden Markov model. In *Advances in Neural Information Processing Systems*, volume 14, 2002.

[8] Y. W. Teh, M. I. Jordan, M. J. Beal, and D. M. Blei. Hierarchical Dirichlet processes. *Journal of the American Statistical Association*, 101(476):1566–1581, 2006.

[9] E. P. Xing and K. Sohn. Hidden Markov Dirichlet process: Modeling genetic recombination in open ancestral space. *Bayesian Analysis*, 2(2), 2007.

[10] R. R. Hudson. Properties of a neutral allele model with intragenic recombination. *Theoretical Population Biology*, 23(2):183 – 201, 1983.

[11] M. Stephens and P. Donnelly. A comparison of Bayesian methods for haplotype reconstruction from population genotype data. *American Journal of Human Genetics*, 73:1162–1169.

[12] N. Li and M. Stephens. Modeling Linkage Disequilibrium and Identifying Recombination Hotspots Using Single-Nucleotide Polymorphism Data. *Genetics*, 165(4):2213–2233, 2003.

[13] J. Marchini, B. Howie, S. Myers, G. McVean, and P. Donnelly. A new multipoint method for genome-wide association studies by imputation of genotypes. *Nature Genetics*, 39(7):906–913, 2007.

[14] B. L. Browning and S. R. Browning. A unified approach to genotype imputation and haplotype-phase inference for large data sets of trios and unrelated individuals. *American Journal of Human Genetics*, 84:210–223, 2009.

[15] D. Aldous. Exchangeability and related topics. In *École d'Été de Probabilités de Saint-Flour XIII–1983*, pages 1–198. Springer, Berlin, 1985.

[16] J. Pitman. *Combinatorial Stochastic Processes*. Lecture Notes in Mathematics. Springer-Verlag, 2006.

[17] D. Blackwell and J. B. MacQueen. Ferguson distributions via Pólya urn schemes. *Annals of Statistics*, 1:353–355, 1973.

[18] E. Çinlar. *Introduction to Stochastic Processes*. Prentice Hall, 1975.

[19] R. M. Neal. Slice sampling. *Annals of Statistics*, 31:705–767, 2003.

[20] J. F. C. Kingman. On the genealogy of large populations. *Journal of Applied Probability*, 19:27–43, 1982. Essays in Statistical Science.

[21] J. F. C. Kingman. The coalescent. *Stochastic Processes and their Applications*, 13:235–248, 1982.

[22] G. A. T. McVean and N. J. Cardin. Approximating the coalescent with recombination. *Philosophical Transactions of the Royal Society of London B: Biological Sciences*, 360(1459):1387–1393, 2005.

[23] P. Marjoram and J. Wall. Fast "coalescent" simulation. *BMC Genetics*, 7(1):16, 2006.

[24] J. Bertoin. *Random Fragmentation and Coagulation Processes*. Cambridge University Press, 2006.

[25] J. Pitman and M. Yor. The two-parameter Poisson-Dirichlet distribution derived from a stable subordinator. *Annals of Probability*, 25:855–900, 1997.

[26] J. Pitman. Coalescents with multiple collisions. *Annals of Probability*, 27:1870–1902, 1999.

[27] J. Gasthaus and Y. W. Teh. Improvements to the sequence memoizer. In *Advances in Neural Information Processing Systems*, 2010.

[28] V. Rao and Y. W. Teh. Fast MCMC sampling for Markov jump processes and continuous time Bayesian networks. In *Proceedings of the International Conference on Uncertainty in Artificial Intelligence*, 2011.

[29] NHLBI Program for Genomic Applications. SeattleSNPs. June 2011. http://pga.gs.washington.edu.

[30] B. N. Howie, P. Donnelly, and J. Marchini. A flexible and accurate genotype imputation method for the next generation of genome-wide association studies. *PLoS Genetics*, (6), 2009.

[31] J. Berestycki. Exchangeable fragmentation-coalescence processes and their equilibrium measures. http://arxiv.org/abs/math/0403154, 2004.

